# Learning in Computer Vision and Image Understanding

**Hayit Greenspan**
Department of Electrical Engineering
California Institute of Technology, 116-81
Pasadena, CA 91125

There is an increasing interest in the area of Learning in Computer Vision and Image Understanding, both from researchers in the learning community and from researchers involved with the computer vision world. The field is characterized by a shift away from the classical, purely model-based, computer vision techniques, towards data-driven learning paradigms for solving real-world vision problems.

Using *learning* in segmentation or recognition tasks has several advantages over classical model-based techniques. These include adaptivity to noise and changing environments, as well as in many cases, a simplified system generation procedure. Yet, learning from examples introduces a new challenge - getting a representative data set of examples from which to learn. Applications of learning systems to practical problems have shown that the performance of the system is often critically dependent on both the size and quality of the training set. **Federico Girosi** of **MIT** suggested the use of prior information as a general method for synthesizing many training examples from few exemplars. Prototypical transformations are used for general 3D object recognition. Face-recognition was presented as a particular example. **Dean Pomerleau** of **Carnegie Mellon** addressed the training data problem as well, within the context of ALVINN, a neural network vision system which drives an autonomous van without human intervention. Some general problems emerge, such as getting sufficient training data for the more unexpected scenes including passing cars and intersections. Several techniques for exploiting prior geometric knowledge during training and testing of the neural-network, were presented. A somewhat different perspective was presented by **Bartlett Mel** of **Caltech**. Bartlett introduced a 3D object recognition approach based on concepts from the human visual system. Here the assumption is that a large database of examples exists, with varying viewing angles and distances, as is available to human observers as they manipulate and inspect common objects.

A different issue of interest was using learning schemes in general recognition frameworks which can handle several different vision problems. **Hayit Greenspan** of **Caltech** suggested combining unsupervised and supervised learning approaches within a multiresolution image representation space, for texture and shape recognition. It was suggested that shifting the input pixel representation to a more robust representation (using a pyramid filtering approach) in combination with learning

schemes can combine the advantages of both approaches. **Jonathan Marshall** of **Univ. of North Carolina** concentrated on *unsupervised* learning and proposed that a common set of unsupervised learning rules might provide a basis for communication between different visual modules (such as stereopsis, motion perception, depth and so forth).

The role of *unsupervised learning* in vision tasks, and its combination with supervised learning, was an issue of discussion. The question arose on how much unsupervised learning is actually *unsupervised*. Some a-priori knowledge, or bias, is always present (e.g., the metric chosen for the task). **Eric Saund** of **Xerox** introduced the window registration problem in unsupervised learning of visual features. He argued that there is a strong dependence on the window placement as slight shifts in the window placement can represent confounding assignments of image data to the input units of the classifying network. **Chris Williams** of **Toronto** introduced the use of unsupervised learning for classifying objects. Given a set of images, each of which contains one instance of a small but unknown set of objects imaged from a random viewpoint, unsupervised learning is used to discover the object classes. Data is grouped into objects via a mixture model which is trained with the EM algorithm.

Real-world computer vision applications in which learning can play a major role, and the challenges involved, was an additional theme in the workshop. **Yann Le Cun** of **AT&T** described a handwritten word recognizer system of multiple modules, as an example of a large scale vision system. Yann suggested that increasing the role of learning in all modules allows one to minimize the amount of hand-built heuristics and improves the robustness and generality of the system. Challenges include training large learning machines which are composed of multiple, heterogeneous modules, and what the modules should contain. **Padhraic Smyth** of **JPL** introduced the challenges for vision and learning in the context of large scientific image databases. In this domain there is often a large amount of data which typically has no ground truth labeling. In addition, natural objects can be much more difficult to deal with than man made objects. Learning can be valuable here, as a low-cost solution and sometimes the *only* solution (with model-based schemes being impractical). The task of face recognition was addressed by **Joachim Buhmann** of **Bonn**. Elastic matching was introduced for translation, rotation and scale invariant recognition. Methods to combine unsupervised and supervised data clustering with elastic matching to learn a discriminant metric and enhance saliency of prototypes were discussed. Related issues from a recent AAAI forum on Machine Learning in Computer Vision, were presented by **Rich Zemel** of the **Salk Institute**.

## In Conclusion
The vision world is very diverse with each different task introducing a whole spectrum of challenges and open issues. Currently, many of the approaches are very application dependent. It is clear that much effort still needs to be put in the definition of the underlying themes of the field as combined across the different application domains. There was general agreement at the workshop that the issues brought up should be pursued further and discussed at future follow-up workshops.

Special thanks to Padhraic Smyth, Tommy Poggio, and Rama Chellappa for their contribution to the organization of the workshop.